# Reinforcement Learning with Soft State Aggregation

**Satinder P. Singh**
singh@psyche.mit.edu

**Tommi Jaakkola**
tommi@psyche.mit.edu

**Michael I. Jordan**
jordan@psyche.mit.edu

Dept. of Brain & Cognitive Sciences (E-10)
M.I.T.
Cambridge, MA 02139

## Abstract

It is widely accepted that the use of more compact representations than lookup tables is crucial to scaling reinforcement learning (RL) algorithms to real-world problems. Unfortunately almost all of the theory of reinforcement learning assumes lookup table representations. In this paper we address the pressing issue of combining function approximation and RL, and present 1) a function approximator based on a simple extension to state aggregation (a commonly used form of compact representation), namely *soft* state aggregation, 2) a theory of convergence for RL with arbitrary, but fixed, soft state aggregation, 3) a novel intuitive understanding of the effect of state aggregation on online RL, and 4) a new heuristic *adaptive* state aggregation algorithm that finds improved compact representations by exploiting the non-discrete nature of soft state aggregation. Preliminary empirical results are also presented.

## 1 INTRODUCTION

The strong theory of convergence available for reinforcement learning algorithms (e.g., Dayan & Sejnowski, 1994; Watkins & Dayan, 1992; Jaakkola, Jordan & Singh, 1994; Tsitsiklis, 1994) makes them attractive as a basis for building learning control architectures to solve a wide variety of search, planning, and control problems. Unfortunately, almost all of the convergence results assume lookup table representa-

tions for value functions (see Sutton, 1988; Dayan, 1992; Bradtke, 1993; and Vanroy & Tsitsiklis, personal communication; for exceptions). It is widely accepted that the use of more compact representations than lookup tables is crucial to scaling RL algorithms to real-world problems.

In this paper we address the pressing issue of combining function approximation and RL, and present 1) a function approximator based on a simple extension to state aggregation (a commonly used form of compact representation, e.g., Moore, 1991), namely *soft* state aggregation, 2) a theory of convergence for RL with arbitrary, but fixed, soft state aggregation, 3) a novel intuitive understanding of the effect of state aggregation on online RL, and 4) a new heuristic *adaptive* state aggregation algorithm that finds improved compact representations by exploiting the non-discrete nature of soft state aggregations. Preliminary empirical results are also presented.

**Problem Definition and Notation:** We consider the problem of solving large Markovian decision processes (MDPs) using RL algorithms and compact function approximation. We use the following notation: $S$ for state space, $A$ for action space, $P^a(s, s')$ for transition probability, $R^a(s)$ for payoff, and $\gamma$ for discount factor. The objective is to maximize the expected, infinite horizon, discounted sum of payoffs.

## 1.1   FUNCTION APPROXIMATION: SOFT STATE CLUSTERS

In this section we describe a new function approximator (FA) for RL. In section 3 we will analyze it theoretically and present convergence results. The FA maps the state space $S$ into $M > 0$ aggregates or clusters from cluster space $\mathcal{X}$. Typically, $M << |S|$. We allow *soft* clustering, where each state $s$ belongs to cluster $x$ with probability $P(x|s)$, called the clustering probabilities. This allows each state $s$ to belong to several clusters. An interesting special case is that of the usual state aggregation where each state belongs only to one cluster. The theoretical model is that the agent can observe the underlying state but can only update a value function for the clusters. The value of a cluster *generalizes* to all states in proportion to the clustering probabilities. Throughout we use the symbols $x$ and $y$ to represent individual clusters and the symbols $s$ and $s'$ to represent individual states.

## 2   A GENERAL CONVERGENCE THEOREM

An online RL algorithm essentially sees a sequence of quadruples, $< s_t, a_t, s_{t+1}, r_t >$, representing a transition from current state $s_t$ to next state $s_{t+1}$ on current action $a_t$ with an associated payoff $r_t$. We will first prove a general convergence theorem for Q-learning (Watkins & Dayan, 1992) applied to a sequence of quadruples that may or may not be generated by a Markov process (Bertsekas, 1987). This is required because the RL problem at the level of the clusters may be *non*-Markovian. Conceptually, the sequence of quadruples can be thought of as being produced by some process that is allowed to modify the sequence of quadruples produced by a Markov process, e.g., by mapping states to clusters. In Section 3 we will specialize the following theorem to provide specific results for our function approximator.

Consider any stochastic process that generates a sequence of random quadruples, $\Psi = \{< x_i, a_i, y_i, r_i >\}_i$, where $x_i, y_i \in Y$, $a_i \in A$, and $r_i$ is a bounded real number. Note that $x_{i+1}$ does not have to be equal to $y_i$. Let $|Y|$ and $|A|$ be finite, and define

indicator variables

$$\chi_i(x, a, y) = \begin{cases} 1 & \text{when } \Psi_i = <x, a, y, .> \text{ (for any } r) \\ 0 & \text{otherwise,} \end{cases}$$

and

$$\chi_i(x, a) = \begin{cases} 1 & \text{when } \Psi_i = <x, a, ., .> \text{ (for any } y, \text{ and any } r) \\ 0 & \text{otherwise.} \end{cases}$$

Define

$$P_{i,j}^a(x, y) = \frac{\sum_{k=i}^{j} \chi_k(x, a, y)}{\sum_{k=i}^{j} \chi_k(x, a)} \quad \text{and} \quad R_{i,j}^a(x) = \frac{\sum_{k=1}^{j} r_k \chi_k(x, a)}{\sum_{k=i}^{j} \chi_k(x, a)}$$

**Theorem 1:** If $\forall \epsilon > 0$, $\exists M_\epsilon < \infty$, such that for all $i \geq 0$, for all $x, y \in Y$, and for all $a \in A$, the following conditions characterize the infinite sequence $\Psi$: with probability $1 - \epsilon$,

$$|P_{i,i+M_\epsilon}^a(x, y) - \bar{P}^a(x, y)| < \epsilon \quad \text{and}$$
$$|R_{i,i+M_\epsilon}^a(x) - \bar{R}^a(x)| < \epsilon, \tag{1}$$

where for all $x$, $a$, and $y$, with probability one $P_{0,\infty}^a(x, y) = \bar{P}^a(x, y)$, and $R_{0,\infty}^a(x) = \bar{R}^a(x)$. Then, online Q-learning applied to such a sequence will converge with probability one to the solution of the following system of equations: $\forall x \in Y$, and $\forall a \in A$,

$$Q(x, a) = \bar{R}^a(x) + \gamma \sum_{y \in Y} \bar{P}^a(x, y) \max_{a' \in A} Q(y, a') \tag{2}$$

**Proof:** Consider the semi-batch version of Q-learning that collects the changes to the value function for $M$ steps before making the change. By assumption, for any $\epsilon$, making $M_\epsilon$ large enough will ensure that with probability $1 - \epsilon$, the sample quantities for the $i^{th}$ batch, $P_{i,i+M_{\epsilon(i)}}^a(x, y)$ and $R_{i,i+M_\epsilon(i)}^a(x)$ are within $\epsilon$ of the asymptotic quantities. In Appendix $A$ we prove that the semi-batch version of Q-learning outlined above converges to the solution of Equation 2 with probability one. The semi-batch proof can be extended to online Q-learning by using the analysis developed in Theorem 3 of Jaakkola *et al.* (1994). In brief, it can be shown that the difference caused by the online updating vanishes in the limit thereby forcing semi-batch Q-learning and online Q-learning to be equal asymptotically. The use of the analysis in Theorem 3 from Jaakkola *et al.* (1994) requires that the learning rate parameters $\alpha$ are such that $\frac{\alpha_t(x)}{max_{t \in M_\epsilon(k)}\alpha_t(x)} \rightarrow 1$ uniformly w.p.1.; $M_\epsilon(k)$ is the $k^{th}$ batch of size $M_\epsilon$. If $\alpha_t(x)$ is non-increasing in addition to satisfying the conventional Q-learning conditions, then it will also meet the above requirement. $\square$

Theorem 1 provides the most general convergence result available for Q-learning (and TD(0)); it shows that for an arbitrary quadruple sequence satisfying the ergodicity conditions given in Equations 1, Q-learning will converge to the solution of *the* MDP constructed with the limiting probabilities $(P_{0,\infty})$ and payoffs $(R_{0,\infty})$. Theorem 1 combines and generalizes the results on hard state aggregation and value iteration presented in Vanroy & Tsitsiklis (personal communication), and on partially observable MDPs in Singh *et al.* (1994).

## 3   RL AND SOFT STATE AGGREGATION

In this section we apply Theorem 1 to provide convergence results for two cases: 1) using Q-learning and our FA to solve MDPs, and 2) using Sutton's (1988) TD(0) and our FA to determine the value function for a fixed policy. As is usual in online RL, we continue to assume that the transition probabilities and the payoff function of the MDP are unknown to the learning agent. Furthermore, being online such algorithms cannot sample states in arbitrary order. In this section, the clustering probabilities $P(x|s)$ are assumed to be fixed.

**Case 1:  Q-learning and Fixed Soft State Aggregation**

Because of function approximation, the domain of the learned Q-value function is constrained to be $\mathcal{X} \times A$ ($\mathcal{X}$ is cluster space). This section develops a "Bellman equation" (e.g., Bertsekas, 1987) for Q-learning at the level of the cluster space. We assume that the agent follows a stationary stochastic policy $\pi$ that assigns to each state a non-zero probability of executing every action in every state. Furthermore, we assume that the Markov chain under policy $\pi$ is ergodic. Such a policy $\pi$ is a *persistently exciting* policy. Under the above conditions $P^\pi(s|x) = \frac{P(x|s)P^\pi(s)}{\sum_{s'} P(x|s')P^\pi(s')}$, where for all $s$, $P^\pi(s)$ is the steady-state probability of being in state $s$.

**Corollary 1:** Q-learning with soft state aggregation applied to an MDP while following a persistently exciting policy $\pi$ will converge with probability one to the solution of the following system of equations: $\forall (x,a) \in (\mathcal{X} \times A)$,

$$Q(x,a) \;=\; \sum_s P^\pi(s|x) \left[ R^a(s) + \gamma \sum_y P^a(s,y) \max_{a'} Q(y,a') \right] \quad (3)$$

and $P^a(s,y) = \sum_{s'} P^a(s,s')P(y|s')$. The Q-value function for the state space can then be constructed via $Q(s,a) = \sum_x P(x|s)Q(x,a)$ for all $(s,a)$.

**Proof:** It can be shown that the sequence of quadruples produced by following policy $\pi$ and independently mapping the current state $s$ to a cluster $x$ with probability $P(x|s)$ satisfies the conditions of Theorem 1. Also, it can be shown that

$$\bar{P}^a(x,y) = \sum_s P^\pi(s|x)P^a(s,y), \text{ and } \bar{R}^a(x) = \sum_s P^\pi(s|x)R^a(s).$$

Note that the Q-values found by clustering are dependent on the sampling policy $\pi$, unlike the lookup table case.

**Case 2:  TD(0) and Fixed Soft State Aggregation**

We present separate results for TD(0) because it forms the basis for policy-iteration-like methods for solving Markov control problems (e.g., Barto, Sutton & Anderson, 1983) — a fact that we will use in the next section to derive adaptive state aggregation methods. As before, because of function approximation, the domain of the learned value function is constrained to be the cluster space $\mathcal{X}$.

**Corollary 2:** TD(0) with soft state aggregation applied to an MDP while following a policy $\pi$ will converge with probability one to the solution of the following system

of equations: $\forall x \in \mathcal{X}$,

$$V(x) = \sum_s P^\pi(s|x) \left[ R^\pi(s) + \gamma \sum_y P^\pi(s, y)V(y) \right] \qquad (4)$$

where again as in Q-learning the value function for the state space can be constructed via $V(s) = \sum_x P(x|s)V(x)$ for all $s$.

**Proof:** Corollary 1 implies Corollary 2 because TD(0) is a special case of Q-learning for MDPs with a single (possibly randomized) action in each state. Equation 4 provides a "Bellman equation" for TD(0) at the level of the cluster space. $\square$

## 4  ADAPTIVE STATE AGGREGATION

In previous sections we restricted attention to a function approximator that had a *fixed* compact representation. How might one adapt the compact representation online in order to get better approximations of value functions? This section presents a novel *heuristic* adaptive algorithm that improves the compact representation by finding good clustering probabilities given an a priori fixed number of clusters. Note that for arbitrary clustering, while Corollaries 1 and 2 show that RL will find solutions with zero Bellman error in cluster space, the associated Bellman error in the state space will *not* be zero in general. Good clustering is therefore naturally defined in terms of reducing the Bellman error for the states of the MDP.

Let the clustering probabilities be parametrized as follows $P(x|s; \theta) = \frac{e^{\theta(x,s)}}{\sum_{x'} e^{\theta(x',s)}}$, where $\theta(x, s)$ is the weight between state $s$ and cluster $x$. Then the Bellman error at.state $s$ given parameter $\theta$ (a matrix) is,

$$J(s|\theta) = V(s|\theta) - \left[ R^\pi(s) + \gamma \sum_{s'} P^\pi(s, s')V(s'|\theta) \right]$$

$$= \sum_x P(x|s; \theta)V(x|\theta) - \left[ R^\pi(s) + \gamma \sum_{s'} P^\pi(s, s') \sum_x P(x|s'; \theta)V(x|\theta) \right]$$

**Adaptive State Aggregation (ASA) Algorithm:**

> Step 1: Compute $V(x|\theta)$ for all $x \in \mathcal{X}$ using the TD(0) algorithm.
> Step 2: Let $\Delta\theta = -\alpha \frac{\partial J^2(\theta)}{\partial \theta}$. Go to step 1.

where Step 2 tries to minimize the Bellman error for the states by holding the cluster values fixed to those computed in Step 1. We have

$$\frac{\partial J^2(s|\theta)}{\partial \theta(y, s)} = 2J(s|\theta) \left[ P(y|s; \theta)(1 - \gamma P^\pi(s, s))(V(y|\theta) - V(s|\theta)) \right].$$

The Bellman error $J(s|\theta)$ cannot be computed directly because the transition probabilities $P(s, s')$ are unknown. However, it can be estimated by averaging the sample

Bellman error. $P(y|s;\theta)$ is known, and $(1 - \gamma P^\pi(s,s))$ is always positive, and independent of $y$, and can therefore be absorbed into the step-size $\alpha$. The quantities $V(y|\theta)$ and $V(s|\theta)$ are available at the end of Step 1. In practice, Step 1 is only carried out partially before Step 2 is implemented. Partial evaluation works well because the changes in the clustering probabilities at Step 2 are small, and because the final $V(x|\theta)$ at the previous Step 1 is used to initialize the computation of $V(x|\theta)$ at the next Step 1.

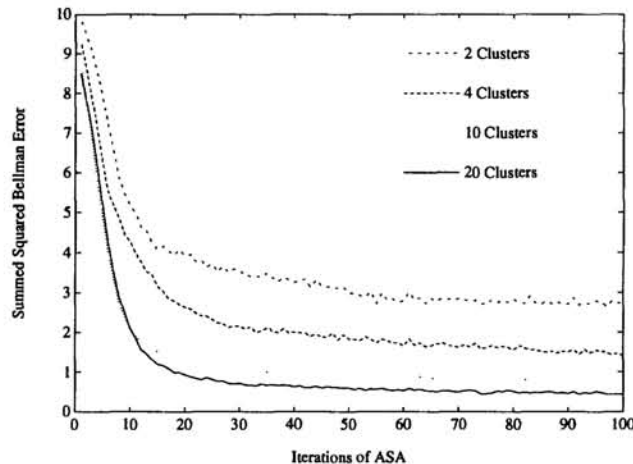

Figure 1: Adaptive State Clustering. See text for explanation.

Figure 1 presents preliminary empirical results for the ASA algorithm. It plots the squared Bellman error summed over the state space as a function of the number of iterations of the ASA algorithm with constant step-size $\alpha$. It shows error curves for 2, 4, 10 and 20 clusters averaged over ten runs of randomly constructed 20 state Markov chains. Figure 4 shows that ASA is able to adapt the clustering probabilities to reduce the Bellman error in state space, and as expected the more clusters the smaller the asymptotic Bellman error. In future work we plan to test the policy iteration version of the adaptive soft aggregation algorithm on Markov control problems.

## 5   SUMMARY AND FUTURE WORK

Doing RL on aggregated states is potentially very advantageous because the value of each cluster generalizes across all states in proportion to the clustering probabilities. The same generalization is also potentially perilous because it can interfere with the contraction-based convergence of RL algorithms (see Yee, 1992; for a discussion). This paper resolves this debate for the case of soft state aggregation by defining a set of Bellman Equations (3 and 4) for the control and policy evaluation problems in the non-Markovian cluster space, and by proving that Q-learning and TD(0) solve them respectively with probability one. Theorem 1 presents a general convergence result that was applied to state aggregation in this paper, but is also a generalization of the results on hidden state presented in Singh *et al.* (1994), and may be applicable

to other novel problems. It supports the intuitive picture that if a non-Markovian sequence of state transitions and payoffs is ergodic in the sense of Equation 1, then RL algorithms will converge w.p.1. to the solution of an MDP constructed with the limiting transition probabilities and payoffs.

We also presented a new algorithm, ASA, for adapting compact representations, that takes advantage of the soft state aggregation proposed here to do gradient descent in clustering probability space to minimize squared Bellman error in the state space. We demonstrated on simple examples that ASA is able to adapt the clustering probabilities to dramatically reduce the Bellman error in state space. In future work we plan to extend the convergence theory presented here to discretizations of continuous state MDPs, and to further test the ASA algorithms.

# A Convergence of semi-batch Q-learning (Theorem 1)

Consider a semi-batch algorithm that collects the changes to the Q-value function for $M$ steps before making the change to the Q-value function. Let

$$R_k^a(x) = \sum_{i=(k-1)M}^{kM} r_i \chi_i(x, a); \quad M_k(x, a) = \sum_{i=(k-1)M}^{kM} \chi_i(x, a)$$

and

$$M_k(x, a, y) = \sum_{i=(k-1)M}^{kM} \chi_i(x, a, y)$$

Then the Q-value of $(x, a)$ after the $k^{th}$ batch is given by:

$$
Q_{k+1}(x, a) = (1 - M_k(x, a)\alpha_k(x, a))Q_k(x, a)
$$
$$
+ M_k(x, a)\alpha_k(x, a) \left[ \frac{R_k^a(x)}{M_k(x, a)} + \gamma \sum_{y \in Y} \frac{M_k(x, a, y)}{M_k(x, a)} \max_{a'} Q_k(y, a') \right]
$$

Let $\bar{Q}$ be the solution to Equation 2. Define,

$$
F_k(x, a) = \frac{R_k^a(x)}{M_k(x, a)} + \gamma \sum_{y \in Y} \frac{M_k(x, a, y)}{M_k(x, a)} \max_{a'} Q_k(y, a') - \bar{Q}(x, a),
$$

then, if $V_k(x) = \max_a Q_k(x, a)$ and $\bar{V}(x) = \max_a \bar{Q}(x, a)$,

$$
F_k(x, a) = \gamma \sum_y \frac{M_k(x, a, y)}{M_k(x, a)}[V_k(y) - \bar{V}(y)] + \left( \frac{R_k^a(x)}{M_k(x, a)} - R_{0,\infty}^a(x) \right)
$$
$$
+ \gamma \sum_y \left[ \left( \frac{M_k(x, a, y)}{M_k(X, a)} - P_{0,\infty}^a(x, y) \right) \bar{V}(y) \right],
$$

The quantity $F_k(x, a)$ can be bounded by

$$\|F_k(x, a)\| \leq \gamma \|V_k - \bar{V}\| + \|(\frac{R_k^a(x)}{M_k(x,a)} - R_{0,\infty}^a(x))\|$$

$$+ \gamma \|\sum_y (\frac{M_k(x,a,y)}{M_k(x,a)} - P_{0,\infty}^a(x, y))\bar{V}(y)\| \leq \gamma \|V_k - \bar{V}\| + C\epsilon_k^M,$$

where $\epsilon_k^M$ is the larger of $|\frac{R_k^a(x)}{M_k(x,a)} - R_{0,\infty}^a(x)|$, and $\gamma |\sum_y (\frac{M_k(x,a,y)}{M_k(x,a)} - P_{0,\infty}^a(x, y))|$. By

assumption for any $\epsilon > 0$, $\exists M_\epsilon < \infty$ such that $\epsilon_k^{M_\epsilon} < \epsilon$ with probability $1 - \epsilon$. The variance of $F_k(x, a)$ can also be shown to be bounded because the variance of the sample probabilities is bounded (everything else is similar to standard Q-learning for MDPs). Therefore by Theorem 1 of Jaakkola *et al.* (1994), for any $\epsilon > 0$, with probability $(1 - \epsilon)$, $Q_k(x, a) \to Q_\infty(x, a)$, where $|Q_\infty(x, a) - \bar{Q}(x, a)| \leq C\epsilon$. Therefore, semi-batch Q-learning converges with probability one. $\square$

## Acknowledgements

This project was supported in part by a grant from the McDonnell-Pew Foundation, by a grant from ATR Human Information Processing Research Laboratories, and by a grant from Siemens Corporation. Michael I. Jordan is a NSF Presidential Young Investigator.

## References

A. G. Barto, R. S. Sutton, & C. W. Anderson. (1983) Neuronlike elements that can solve difficult learning control problems. *IEEE SMC*, 13:835–846.

D. P. Bertsekas. (1987) *Dynamic Programming: Deterministic and Stochastic Models*, Prentice-Hall.

S. J. Bradtke. (1993) Reinforcement learning applied to linear quadratic regulation. In *Advances in Neural Information Processing Systems 5*, pages 295–302.

P. Dayan. (1992) The convergence of TD($\lambda$) for general $\lambda$. *Machine Learning*, 8(3/4):341–362.

P. Dayan & T.J. Sejnowski. (1994) TD($\lambda$) converges with probability 1. *Machine Learning*, 13(3).

T. Jaakkola, M. I. Jordan, & S. P. Singh. (1994) On the convergence of stochastic iterative dynamic programming algorithms. *Neural Computation*, 6(6):1185–1201.

A. W. Moore. (1991) Variable resolution dynamic programming: Efficiently learning action maps in multivariate real-valued state-spaces. In *Maching Learning: Proceedings of the Eighth International Workshop*, pages 333–337.

S. P. Singh, T. Jaakkola, & M. I. Jordan. (1994) Learning without state-estimation in partially observable markovian decision processes. In *Machine Learning: Proceedings of the Eleventh International Conference*, pages 284–292.

R. S. Sutton. (1988) Learning to predict by the methods of temporal differences. *Machine Learning*, 3:9–44.

J. Tsitsiklis. (1994) Asynchronous stochastic approximation and Q-learning. *Machine Learning*, 16(3):185–202.

B. Vanroy & J. Tsitsiklis. (personal communication)

C. J. C. H. Watkins & P. Dayan. (1992) Q-learning. *Machine Learning*, 8(3/4):279–292.

R. C. Yee. (1992) Abstraction in control learning. Technical Report COINS Technical Report 92-16, Department of Computer and Information Science, University of Massachusetts, Amherst, MA 01003. A dissertation proposal.